# Occlusion Detection and Motion Estimation with Convex Optimization

**Alper Ayvaci,    Michalis Raptis,    Stefano Soatto**

University of California, Los Angeles
{ayvaci, mraptis, soatto}@cs.ucla.edu

## Abstract

We tackle the problem of simultaneously detecting occlusions and estimating optical flow. We show that, under standard assumptions of Lambertian reflection and static illumination, the task can be posed as a *convex minimization* problem. Therefore, the solution, computed using efficient algorithms, is guaranteed to be *globally optimal*, for any number of independently moving objects, and any number of occlusion layers. We test the proposed algorithm on benchmark datasets, expanded to enable evaluation of occlusion detection performance.

## 1   Introduction

*Optical flow* refers to the deformation of the domain of an image that results from ego- or scene motion. It is, in general, different from the *motion field*, that is the projection onto the image plane of the spatial velocity of the scene [28], unless three conditions are satisfied: (a) Lambertian reflection, (b) constant illumination, and (c) constant visibility properties of the scene. Most surfaces with benign reflectance properties (diffuse/specular) can be approximated as Lambertian almost everywhere under sparse illuminants (e.g., the sun). In any case, widespread violation of Lambertian reflection does not enable correspondence [23], so we will embrace (a) as customary. Similarly, (b) constant illumination is a reasonable assumption for ego-motion (the scene is not moving relative to the light source), and even for objects moving (slowly) relative to the light source.[1] Assumption (c) is the most critical, as it is needed for the motion field to be defined.[2] It is often taken for granted in the optical flow literature, because in the limit where two images are sampled infinitesimally close in time, *there are no occluded regions,* and one can focus solely on motion discontinuities. Thus, most variational motion estimation approaches provide an estimate of a dense flow field at each location on the image domain, *including occluded regions.* Alas, in occluded regions, the problem is *not* that optical flow is discontinuous, or forward-backward inconsistent; it is simply not defined. Motion in occluded regions can be *hallucinated*; However, whatever motion is assigned to an occluded region *cannot be validated from the data.* In defense of these methods, it can be argued that, even without taking the limit, for small parallax (slow-enough motion, or far-enough objects, or fast-enough temporal sampling) occluded areas are small. However, small does not mean unimportant, as occlusions are critical to perception [8] and a key for developing representations for recognition [22].

For this reason, *we focus on issues of visibility in optical flow computation.* We show that forgoing assumption (c) and explicitly representing occlusions is not only conceptually correct, but also algorithmically advantageous, for the resulting optimization problem can be shown to become *convex* once occlusions are explicitly modeled. Therefore, one can guarantee convergence to a globally

optimal solution regardless of initial conditions (sect. 2). We adapt Nesterov's efficient optimization scheme to our problem (sect. 3), and test the resulting algorithm on benchmark datasets (sect. 4), including evaluation of occlusion detection (sect. 1.2).

## 1.1 Related Work

The most common approach to handling occlusions in the optical flow literature is to define them as regions where forward and backwards motion estimates are inconsistent [19, 1]. Most approaches return estimates of motion in the occluded regions, where they cannot be *invalidated:* As we have already pointed out, in an occluded region one cannot determine a motion field that maps one image onto another, because the scene is not visible in one of the two. Some approaches [11, 4], while also exploiting motion symmetry, discount occlusions by weighting the data fidelity with a monotonically decreasing function. The resulting problem is non-convex, and therefore the proposed alternating minimization techniques can be prone to local minima. An alternate approach [15, 14, 25] is to formulate joint motion estimation and occlusion detection in a discrete setting, where it is NP-hard. Various approximate solutions using combinatorial optimization require fine quantization and, therefore, suffer from a large number of labels which results in loose approximation bounds. Another class of methods uses the motion estimation residual to classify a location as occluded or visible wither with a direct threshold on the residual [30] or with a more elaborate probabilistic model [24]. In each case, the resulting optimization is non-convex.

## 1.2 Evaluation

Optical flow estimation is a mature area of computer vision, and benchmark datasets have been developed, *e.g.,* [2]. Unfortunately, no existing benchmark provides ground truth for occluded regions, nor a scoring mechanism to evaluate occlusion detection performance. Motion estimates are scored even in the occluded regions, where the data does not support them. Since our primary goal is to detect occlusions, we have produced a new benchmark by taking a subset of the training data in the Middlebury dataset, and hand-labeled occluded regions. We then use the same evaluation method of the Middlebury *for the (ground truth) regions that are co-visible* in at least two images. This provides a motion estimation score. Then, we provide a separate score for occlusion detection, in terms of precision-recall curves.

## 2 Joint Occlusion Detection and Optical Flow Estimation

In this section, we show how the assumptions (a)-(b) can be used to formulate occlusion detection and optical flow estimation as a joint optimization problem. Let $I : D \subset \mathbb{R}^2 \times \mathbb{R}^+ \to \mathbb{R}^+; \ (x,t) \mapsto I(x,t)$ be a grayscale time-varying image defined on a domain $D$. Under the assumptions (a)-(b), the relation between two consecutive frames in a video $\{I(x,t)\}_{t=0}^T$ is given by

$$I(x,t) = \begin{cases} I(w(x,t), t+dt) + n(x,t), & x \in D \backslash \Omega(t; dt) \\ \rho(x,t), & x \in \Omega(t; dt) \end{cases} \tag{1}$$

where $w : D \times \mathbb{R}^+ \to \mathbb{R}^2; x \mapsto w(x,t) \doteq x + v(x,t)$ is the domain deformation mapping $I(x,t)$ onto $I(x, t+dt)$ everywhere *except at occluded regions*. Usually *optical flow* denotes the incremental displacement $v(x,t) \doteq w(x,t) - x$. The occluded region $\Omega$ can change over time depending on the temporal sampling interval $dt$ and is not necessarily simply-connected; so even if we call $\Omega$ the occluded region (singular), it is understood that it can be made of several disconnected portions. Inside $\Omega$, the image can take any value $\rho : \Omega \times \mathbb{R}^+ \to \mathbb{R}^+$ that is in general unrelated to $I(w(x), t+dt)_{|x \in \Omega}$. In the limit $dt \to 0$, $\Omega(t; dt) = \emptyset$. Because of (almost-everywhere) continuity of the scene and its motion (i), and because the additive term $n(x,t)$ compounds the effects of a large number of independent phenomena[3] and therefore we can invoke the Law of Large Numbers (ii), in general we have that

$$\text{(i)} \ \lim_{dt \to 0} \Omega(t; dt) = \emptyset, \quad \text{and} \quad \text{(ii)} \ n \overset{IID}{\sim} \mathcal{N}(0, \lambda) \tag{2}$$

i.e., the additive uncertainty is normally distributed in space and time with an isotropic and small variance $\lambda > 0$. We define the residual $e : D \to \mathbb{R}$ on the entire image domain $x \in D$, via

$$e(x, t; dt) \doteq I(x, t) - I(w(x, t), t + dt) = \begin{cases} n(x, t), & x \in D\backslash\Omega \\ \rho(x, t) - I(w(x, t), t + dt), & x \in \Omega \end{cases} \quad (3)$$

which we can write as the sum of two terms, $e_1 : D \to \mathbb{R}$ and $e_2 : D \to \mathbb{R}$, also defined on the entire domain $D$ in such a way that

$$\begin{cases} e_1(x, t; dt) \doteq \rho(x, t) - I(w(x, t), t + dt), & x \in \Omega \\ e_2(x, t; dt) \doteq n(x, t), & x \in D\backslash\Omega. \end{cases} \quad (4)$$

Note that $e_2$ is undefined in $\Omega$, and $e_1$ is undefined in $D\backslash\Omega$, in the sense that they can take any value there, including zero, which we will assume henceforth. We can then write, for any $x \in D$,

$$I(x, t) = I(w(x, t), t + dt) + e_1(x, t; dt) + e_2(x, t; dt) \quad (5)$$

and note that, because of (i) $e_1$ is *large but sparse*,[4] while because of (ii) $e_2$ is *small but dense*[4]. We will use this as an inference criterion for $w$, seeking to optimize a data fidelity term that minimizes the number of nonzero elements of $e_1$ (a proxy of the area of $\Omega$), and the negative log-likelihood of $n$.

$$\psi_{\text{data}}(w, e_1) \doteq \|e_1\|_{\mathbb{L}^0(D)} + \frac{1}{\lambda}\|e_2\|_{\mathbb{L}^2(D)} \quad \text{subject to (5)} \quad (6)$$

$$= \frac{1}{\lambda}\|I(x, t) - I(w(x, t), t + dt) - e_1\|_{\mathbb{L}^2(D)} + \|e_1\|_{\mathbb{L}^0(D)}$$

where $\|f\|_{\mathbb{L}^0(D)} \doteq |\{x \in D | f(x) \neq 0\}|$ and $\|f\|_{\mathbb{L}^2(D)} \doteq \int_D |f(x)|^2 dx$. Unfortunately, we do not know anything about $e_1$ other than the fact that it is sparse, and that what we are looking for is $\chi(\Omega) \propto e_1$, where $\chi : D \to \mathbb{R}^+$ is the characteristic function that is non-zero when $x \in \Omega$, i.e., where the occlusion residual is non-zero. So, the data fidelity term depends on $w$ *but also on the characteristic function of the occlusion domain $\Omega$.*[5] For a sufficiently small $dt$, we can approximate, for any $x \in D\backslash\Omega$,

$$I(x, t + dt) = I(x, t) + \nabla I(x, t)v(x, t) + n(x, t) \quad (9)$$

where the linearization error has been incorporated into the uncertainty term $n(x, t)$. Therefore, following the same previous steps, we have

$$\boxed{\psi_{\text{data}}(v, e_1) = \|\nabla I v + I_t - e_1\|_{\mathbb{L}^2(D)} + \lambda\|e_1\|_{\mathbb{L}^0(D)}.} \quad (10)$$

Since we typically do not know the variance $\lambda$ of the process $n$, we will treat it as a tuning parameter, and because $\psi_{\text{data}}$ or $\lambda\psi_{\text{data}}$ yield the same minimizer, we have attributed the multiplier $\lambda$ to the second term. In addition to the data term, because the unknown $v$ is infinite-dimensional and the problem is ill-posed, we need to impose regularization, for instance by requiring that the total variation (TV) be small

$$\psi_{\text{reg}}(v) = \mu\|v_1\|_{TV} + \mu\|v_2\|_{TV} \quad (11)$$

where $v_1$ and $v_2$ are the first and second components of the optical flow $v$, $\mu$ is a multiplier factor to weight the strength of the regularizer and the weighted isotropic TV norm is defined by

$$\|f\|_{TV(D)} = \int_D \sqrt{(g_1(x)\nabla_x f(x))^2 + (g_2(x)\nabla_y f(x))^2} dx,$$

$$\nabla I(x, t) \doteq \left[ \begin{array}{c} I\left(x + \left[\begin{array}{c} 1 \\ 0 \end{array}\right], t\right) - I(x, t) \\ I\left(x + \left[\begin{array}{c} 0 \\ 1 \end{array}\right], t\right) - I(x, t) \end{array} \right]^T \quad (7)$$

$$I_t(x, t) \doteq I(x, t + dt) - I(x, t). \quad (8)$$

where $g_1(x) \approx exp(-\beta|\nabla_x I(x)|)$ and $g_2(x) \approx exp(-\beta|\nabla_y I(x)|)$; $\beta$ is a normalizing factor. TV is desirable in the context of occlusion detection because it does not penalize motion discontinuities significantly. The overall problem can then be written as the minimization of the cost functional $\psi = \psi_{\text{data}} + \psi_{\text{reg}}$, which is

$$\hat{v}_1, \hat{v}_2, \hat{e}_1 = \arg \min_{v_1, v_2, e_1} \underbrace{\|\nabla I v + I_t - e_1\|_{\mathbb{L}^2(D)}^2 + \lambda\|e_1\|_{\mathbb{L}^0(D)} + \mu\|v_1\|_{TV(D)} + \mu\|v_2\|_{TV(D)}}_{\psi(v_1, v_2, e_1)}$$

(12)

In a digital image, the domain $D$ is quantized into an $M \times N$ lattice $\Lambda$, so we can write (12) in matrix form as:

$$\hat{v}_1, \hat{v}_2, \hat{e}_1 = \arg \min_{v_1, v_2, e_1} \frac{1}{2}\|A[v_1, v_2, e_1]^T + b\|_{\ell_2}^2 + \lambda\|e_1\|_{\ell_0} + \mu\|v_1\|_{TV} + \mu\|v_2\|_{TV} \qquad (13)$$

where $e_1 \in \mathbb{R}^{MN}$ is the vector obtained from stacking the values of $e_1(x,t)$ on the lattice $\Lambda$ on top of one another (column-wise), and similarly with the vector field components $\{v_1(x,t)\}_{x \in \Lambda}$ and $\{v_2(x,t)\}_{x \in \Lambda}$ stacked into $MN$-dimensional vectors $v_1, v_2 \in \mathbb{R}^{MN}$. The spatial derivative matrix $A$ is given by $A = [diag(\nabla_x I) \quad diag(\nabla_y I) \quad -\mathcal{I}]$, where $\mathcal{I}$ is the $MN \times MN$ identity matrix, and the temporal derivative values $\{I_t(x,t)\}_{x \in \Lambda}$ are stacked into $b$. For finite-dimensional vectors $u \in \mathbb{R}^{MN}$, $\|u\|_{\ell^2} = \sqrt{\langle u, u \rangle}$, $\|u\|_{\ell_0} = |\{u_i | u_i \neq 0\}|$ and $\|u\|_{TV} = \sum \sqrt{((g_1)_i(u_{i+1} - u_i))^2 + ((g_2)_i(u_{i+M} - u_i))^2}$ where $g_1$ and $g_2$ are the stacked versions of $\{g_1(x)\}_{x \in \Lambda}$ and $\{g_2(x)\}_{x \in \Lambda}$.

In practice, (13) is NP-hard. Therefore, as customary, we relax it by minimizing the *weighted-$\ell_1$* norm of $e_1$, instead of $\ell_0$, such that

$$\hat{v}_1, \hat{v}_2, \hat{e}_1 = \arg \min_{v_1, v_2, e_1} \frac{1}{2}\|A[v_1, v_2, \ e_1]^T + b\|_{\ell_2}^2 + \lambda\|We_1\|_{\ell_1} + \mu\|v_1\|_{TV} + \mu\|v_2\|_{TV} \qquad (14)$$

where $W$ is a diagonal weight matrix and $\|u\|_{\ell_1} = \sum |u_i|$. When $W$ is the identity, (14) becomes a standard convex relaxation of (13) and its *globally optimal solution* can be reached efficiently [27]. However, the $\ell_0$ norm can also be approximated by reweighting $\ell_1$, as proposed by Candes et al. [5], by setting the diagonal elements of $W$ to $w_i \approx 1/(|(e_1)_i| + \epsilon)$, $\epsilon$ small, after each iteration of (14). The data term of the standard (unweighted) relaxation of (13) can be interpreted as a Huber norm [10]. We favor the more general (14) as the resulting estimate of $e_1$ is more stable and sparse.

The model (9) is valid to the extent in which $dt$ is sufficiently small relative to $v$ (or $v$ sufficiently slow relative to $dt$), so the linearization error does not alter the statistics of the residual $n$. When this is not the case, remedies must be enacted to restore *proper sampling* conditions [22] and therefore differentiate contributions to the residual coming from sampling artifacts (aliasing), rather than occlusions. This can be done by solving (14) in scale-space, as customary, with coarser scales used to initialize $\hat{v}_1, \hat{v}_2$ so the increment is properly sampled, and the occlusion term $e_1$ added at the finest scale.

The residual term $e_1$ in (5) have been characterized in some literature as modeling *illumination changes* [21, 16, 26, 13]. Note that, even if the model (5) appears similar, the *priors* on $e_1$ are rather different: Sparsity in our case, smoothness in theirs. While sparsity is clearly motivated by (i), for illumination changes to be properly modeled, a *reflectance function* is necessary, which is absent in all models of the form (5) (see [23].)

## 3 Optimization with Nesterov's Algorithm

In this section, we describe an efficient algorithm to solve (14) based on Nesterov's first order scheme [17] which provides $O(1/k^2)$ convergence in $k$ iterations, whereas for standard gradient descent, it is $O(1/k)$, a considerable advantage for a large scale problem such as (14). To simplify the notation we let $(e_1)_i \doteq w_i(e_1)_i$, so that $A \doteq [diag(\nabla_x I) \quad diag(\nabla_y I) \quad -W^{-1}]$. We then have

> **Initialize** $v_1^0, v_2^0, e_1^0$. For $k \geq 0$
>
> 1. Compute $\nabla\psi(v_1^k, v_2^k, e_1^k)$
>
> 2. Compute $\alpha_k = 1/2(k+1), \tau_k = 2/(k+3)$
>
> 3. Compute $y_k = [v_1^k, v_2^k, e_1^k]^T - (1/L)\nabla\psi(v_1^k, v_2^k, e_1^k)$,
>
> 4. Compute $z_k = [v_1^0, v_2^0, e_1^0]^T - (1/L)\sum_{i=0}^{k}\alpha_i\nabla\psi(v_1^i, v_2^i, e_1^i)$,
>
> 5. Update $[v_1^k, v_2^k, e_1^k]^T = \tau_k z_k + (1-\tau_k)y_k$.
>
> **Stop** when the solution converges.

In order to implement this scheme, we need to address the nonsmooth nature of $\ell_1$ in the computation of $\nabla\psi$ [18], a common problem in sparse optimization [3]. We write $\psi(v_1, v_2, e_1)$ as

$$\psi(v_1, v_2, e_1) = \psi_1(v_1, v_2, e_1) + \lambda\psi_2(e_1) + \mu\psi_3(v_1) + \mu\psi_4(v_2),$$

and compute the gradient of each term separately. $\nabla_{v_1, v_2, e_1}\psi_1(v_1, v_2, e_1)$ is straightforward:

$$\nabla_{v_1, v_2, e_1}\psi_1(v_1, v_2, e_1) = A^T A[v_1, v_2, e_1]^T + A^T b.$$

The other three terms require smoothing. $\psi_2(e_1) = \|e_1\|_{\ell_1}$ can be rewritten as $\psi_2(e_1) = \max_{\|u\|_\infty \leq 1}\langle u, e_1\rangle$ in terms of its conjugate. [18] proposes a smooth approximation

$$\psi_2^\sigma(e_1) = \max_{\|u\|_\infty \leq 1}\langle u, e_1\rangle - \frac{1}{2}\sigma\|u\|_{\ell_2}^2, \tag{15}$$

and shows that (15) is differentiable and $\nabla_{e_1}\psi_2^\sigma(e_1) = u^\sigma$, where $u^\sigma$ is the solution of (15):

$$u_i^\sigma = \begin{cases} \sigma^{-1}(e_1)_i, & |(e_1)_i| < \sigma, \\ \mathrm{sgn}((e_1)_i), & \text{otherwise.} \end{cases} \tag{16}$$

Following [3], $\nabla_{v_1}\psi_3$ is given by $\nabla_{v_1}\psi_3^\sigma(v_1) = G^T u^\sigma$ where $G = [G_1, G_2]^T$, $G_1$ and $G_2$ are weighted horizontal and vertical differentiation operators, and $u^\sigma$ has the form $[u^1, u^2]$ where

$$u_i^{1,2} = \begin{cases} \sigma^{-1}(G_{1,2}v_1)_i, & \|[(G_1 v_1)_i \ (G_2 v_1)_i]^T\|_{\ell_2} < \sigma, \\ \|[(G_1 v_1)_i \ (G_2 v_1)_i]^T\|_{\ell_2}^{-1}(G_{1,2}v_1)_i, & \text{otherwise.} \end{cases} \tag{17}$$

$\nabla_{v_2}\psi_4$ can be computed in the same way. Once we have computed each term, $\nabla\psi(v_1, v_2, e_1)$ is

$$\nabla\psi(v_1, v_2, e_1) = \nabla\psi_1 + [\lambda\nabla_{e_1}\psi_2, \mu\nabla_{v_1}\psi_3, \mu\nabla_{v_2}\psi_4]^T. \tag{18}$$

We also need the Lipschitz constant $L$ to compute the auxiliary variables $y_k$ and $z_k$ to minimize $\psi$. Since $\|G^T G\|_2$ is bounded above [7] by 8, given the coefficients $\lambda$ and $\mu$, $L$ is given by

$$L = \max(\lambda, 8\mu)/\sigma + \|A^T A\|_2.$$

A crucial element of the scheme is the selection of $\sigma$. It trades off accuracy and speed of convergence. A large $\sigma$ yields a smooth solution, which is undesirable when minimizing the $\ell_1$ norm. A small $\sigma$ causes slow convergence. We have chosen $\sigma$ empirically, although the continuation algorithm proposed in [3] could be employed to adapt $\sigma$ during convergence.

## 4 Experiments

To evaluate occlusion detection (Sect. 1.2), we start from [2] and generate occlusion maps as follows: for each training sequence, the residual computed from the given ground truth motion is used as a discriminant to determine ground truth occlusions, fixing obvious errors in the occlusion maps by hand. We therefore restrict the evaluation of motion to the co-visible regions, and evaluate occlusion detection as a standard binary classification task. We compare our algorithm to [29] and [14], the former is an example of robust motion estimation and the latter is a representative of the approaches described in Sect. 1.1.

In our implementation[6], we first solve (14) with standard relaxation ($W$ is the identity) and then with reweighted-$\ell_1$. To handle large motion, we use a pyramid with scale factor 0.5 and up to 4 levels; $\lambda$ and $\mu$ are fixed at 0.002 and 0.001 (Flower Garden) and 0.0006 and 0.0003 (Middlebury) respectively. To make comparison with [29] fair, we modify the code provided online[7] to include

anisotropic regularization (Fig. 1). Note that no occlusion is present in the residual of the motion field computed by TV-L1, and subsequently the motion estimates are less precise around occluding boundaries (top-left corner of the Flower Garden, plane in the left in Venus).

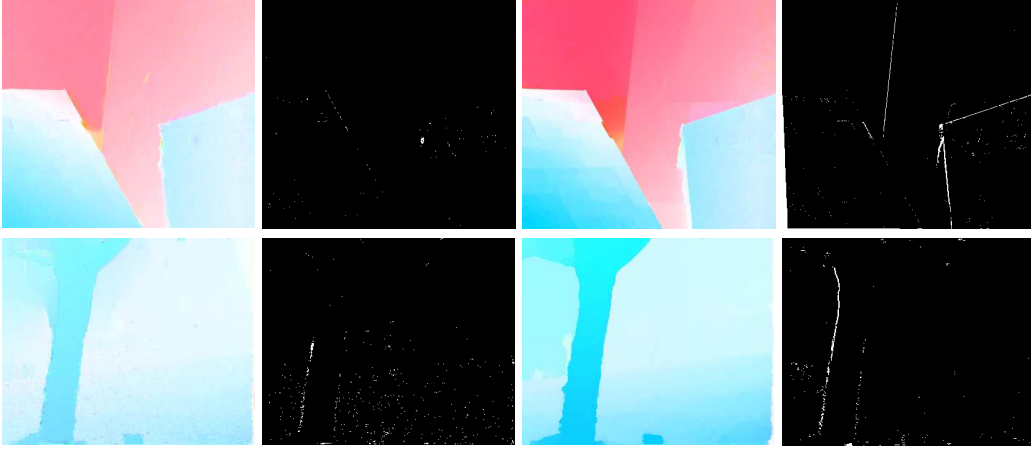

Figure 1: *Comparison with TV-L1 [29] on "Venus" from [2] and "Flower Garden." The first column shows the motion estimates by TV-L1, color-coded as in [29], the second its residual $I(x,t) - I(w(x), t+dt)$; the third shows our motion estimates, and the fourth our residual $e_1$ defined in (14).*

Other frames of the Flower Garden sequence are shown in Fig. 2, where we have regularized the occluded region by minimizing a unilateral energy on $e_1$ with graph-cuts. We have also compared

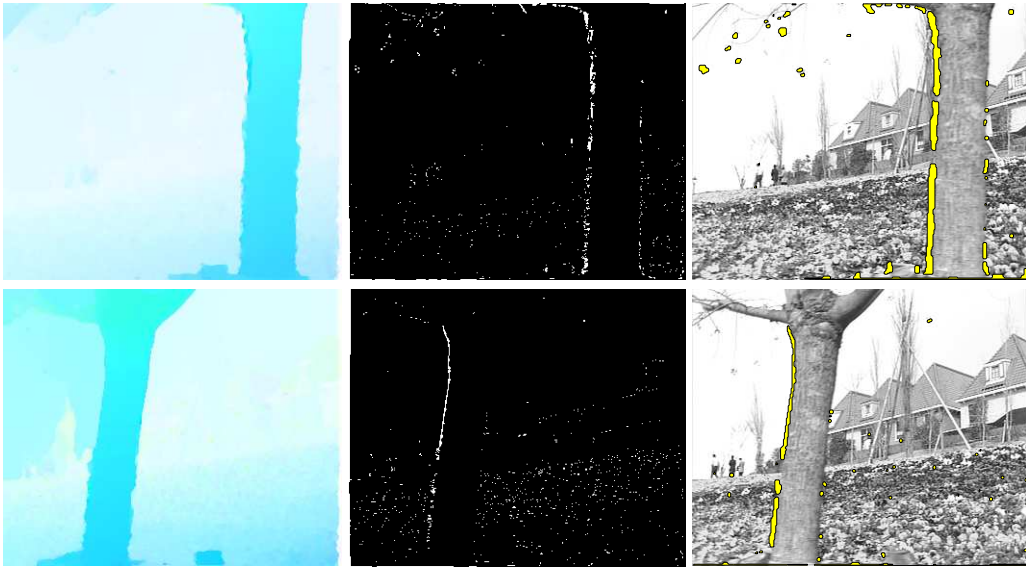

Figure 2: *Motion estimates for more frames of the Flower Garden sequence (left), residual $e$ (middle), and occluded region (right).*

motion estimates obtained with our method and [29] in the co-visible regions for the Middlebury dataset (Table 1). Since occlusions can only be determined at the finest scale absent proper sampling conditions, in this experiment we minimize the same functional of [29] at coarse scales, and switch to (14) at the finest scale. To evaluate occlusion detection performance, we again use the Middlebury, and compare $e_1$ to ground truth occlusions using precision/recall curves (Fig. 3) and average precision values (Table 2). We also show the improvement in detection performance when we use reweighted-$\ell_1$, in Table 2. We have compared our occlusion detection results to [14], using the code provided online by the authors (Table 3). Comparing motion estimates gives an unfair

| | Venus | RubberWhale | Hydrangea | Grove2 | Grove3 | Urban2 | Urban3 |
|---|---|---|---|---|---|---|---|
| AAE (ours) | **4.37** | 5.42 | **2.35** | **2.32** | **5.72** | 3.60 | **6.41** |
| AAE (L1TV) | 5.28 | **4.49** | 2.44 | 3.45 | 7.66 | **3.57** | 7.12 |
| AEPE (ours) | **0.30** | 0.18 | **0.19** | **0.16** | **0.59** | **0.39** | **0.84** |
| AEPE (L1TV) | 0.33 | **0.13** | 0.20 | 0.24 | 0.74 | 0.46 | 0.89 |

Table 1: *Quantitative comparison of our algorithm with TV-L1 [29]. Average Angular Error (AAE) and Average End Point Error (AEPE) of motion estimates in co-visible regions.*

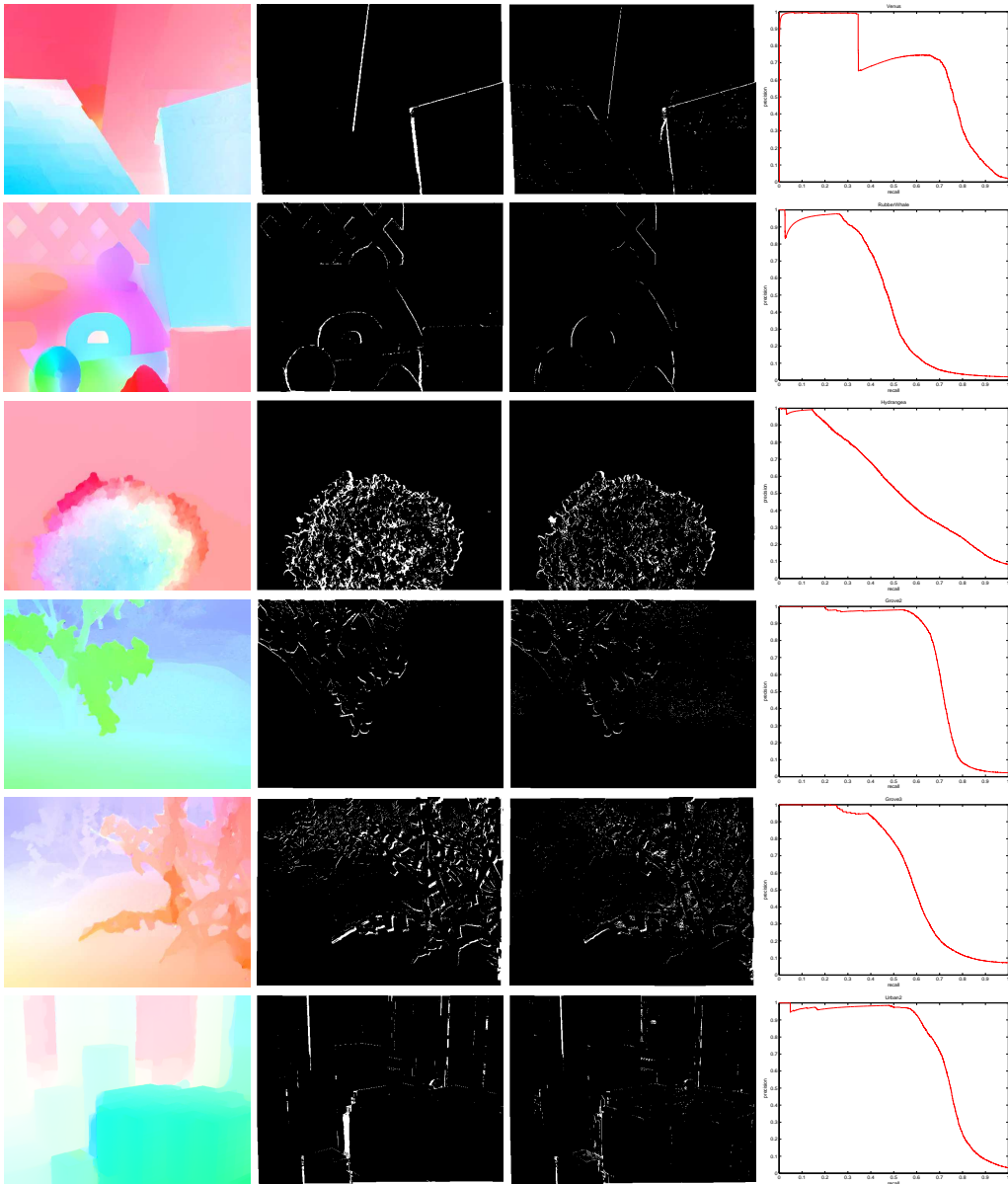

Figure 3: *Left to right: Representative samples of motion estimates from the Middlebury dataset, labeled ground-truth occlusions, error term estimate $e_1$, and precision-recall curves for our occlusion detection.*

advantage to our algorithm because their approach is based on quantized disparity values, yielding lower accuracy.

|  | Venus | Rubber Whale | Hydrangea | Grove2 | Grove3 | Urban2 | Urban3 |
|---|---|---|---|---|---|---|---|
| $\ell_1$ | 0.67 | 0.48 | 0.55 | 0.70 | 0.60 | 0.72 | 0.80 |
| reweighted-$\ell_1$ | 0.69 | 0.49 | 0.57 | 0.70 | 0.61 | 0.73 | 0.80 |

Table 2: *Average precision of our approach on Middlebury data with and without re-weighting.*

It takes 186 seconds for a Matlab/C++ implementation of Nesterov's algorithm to converge to a solution on a $288 \times 352$ frame from Flower Garden sequence. We have also compared Nesterov's algorithm to split-Bregman's method [9] for minimization of (14) in terms of convergence speed and reported the results in [20].

|  | Venus | RubberWhale | Hydrangea | Grove2 | Grove3 | Urban2 | Urban3 |
|---|---|---|---|---|---|---|---|
| Precision [14] | 0.61 | 0.46 | 0.68 | 0.72 | 0.79 | 0.26 | 0.56 |
| Recall [14] | 0.66 | 0.20 | 0.20 | 0.55 | 0.45 | 0.50 | 0.51 |
| Precision(ours) | **0.69** | **0.91** | **0.96** | **0.96** | **0.86** | **0.95** | **0.94** |

Table 3: *Comparison with [14] on Middlebury. Since Kolmogorov et al. provide a binary output, we display our precision at their same recall value.*

## 5 Discussion

We have presented an algorithm to detect occlusions and establish correspondence between two images. It leverages on a formulation that, starting from standard assumptions (Lambertian reflection, constant diffuse illumination), arrives at a *convex optimization problem*. Our approach does *not* assume a rigid scene, nor a single moving object. It also does *not* assume that the occluded region is simply connected: Occlusions in natural scenes can be very complex (see Fig. 3) and should therefore, in general, not be spatially regularized. The fact that occlusion detection reduces to a two-phase segmentation of the domain into either occluded ($\Omega$) or visible ($D\backslash\Omega$) should not confuse the reader familiar with the *image segmentation* literature whereby two-phase segmentation of *one object* (foreground) from the background can be posed as a convex optimization problem [6], but breaks down in the presence of multiple objects, or "phases." Note that in [6] the problem can be made convex only in $e_1$, but not jointly in $e_1$ and $v$. We focus on inter-frame occlusion detection; temporal consistency of occlusion "layers" was addressed in [12].

The limitations of our approach stand mostly in its dependency from the regularization coefficients $\lambda$ and $\mu$. In the absence of some estimate of the variance coefficient $\lambda$, one is left with tuning it by trial-and-error. Similarly, $\mu$ is a parameter that, like in any classification problem, trades off missed detections and false alarms, and therefore no single value is "optimal" in any meaningful sense. These limitations are shared by most variational optical flow estimation algorithms.

**Acknowledgement**: This work was supported by AFOSR FA9550-09-1-0427, ARO 56765-CI, and ONR N00014-08-1-0414.

## Footnotes

[1] Assumption (b) is also made for convenience, as modeling illumination changes would require modeling reflectance, which significantly complicates the picture.

[2] If the domain of an image portrays a portion of the scene that is *not visible* in another image, the two can*not* be put into correspondence.

[3]$n(x,t)$ collects all unmodeled phenomena including deviations from Lambertian reflection, illumination changes, quantization error, sensor noise, and later also linearization error. It does *not* capture occlusions, since those are explicitly modeled.

[4]*Sparse* stands for almost everywhere zero on $D$. Similarly, *dense* stands for almost everywhere non-zero.

[5]In a digital image, both domains $D$ and $\Omega$ are discretized into a lattice, and $dt$ is fixed. Therefore, spatial and temporal derivative operators are approximated, typically, by first-order differences. We use the formal notation

[6]The source code is available at http://vision.ucla.edu/~ayvaci/occlusion-detection/

[7]http://gpu4vision.icg.tugraz.at

## References

[1] L. Alvarez, R. Deriche, T. Papadopoulo, and J. Sánchez. Symmetrical dense optical flow estimation with occlusions detection. *International Journal of Computer Vision*, 75(3):371–385, 2007.

[2] S. Baker, D. Scharstein, J. Lewis, S. Roth, M. Black, and R. Szeliski. A database and evaluation methodology for optical flow. In *Proceedings of the International Conference on Computer Vision*, volume 5, 2007.

[3] S. Becker, J. Bobin, and E. Candes. Nesta: A fast and accurate first-order method for sparse recovery. *Arxiv preprint arXiv*, 904, 2009.

[4] R. Ben-Ari and N. Sochen. Variational stereo vision with sharp discontinuities and occlusion handling. *ICCV. IEEE Computer Society*, pages 1–7, 2007.

[5] E. Candes, M. Wakin, and S. Boyd. Enhancing sparsity by reweighted 1 minimization. *Journal of Fourier Analysis and Applications*, 14(5):877–905, 2008.

[6] T. Chan, S. Esedoglu, and M. Nikolova. Algorithms for finding global minimizers of denoising and segmentation models. *SIAM J. Appl. Math*, 66(1632-1648):1, 2006.

[7] J. Dahl, P. Hansen, S. Jensen, and T. Jensen. Algorithms and software for total variation image reconstruction via first-order methods. *Numerical Algorithms*, pages 67–92, 2009.

[8] J. J. Gibson. *The ecological approach to visual perception*. LEA, 1984.

[9] T. Goldstein and S. Osher. The split Bregman method for L1 regularized problems. *SIAM Journal on Imaging Sciences*, 2(2):323–343, 2009.

[10] P. Huber and E. Ronchetti. *Robust statistics*. John Wiley & Sons Inc, 2009.

[11] S. Ince and J. Konrad. Occlusion-aware optical flow estimation. *IEEE Transactions on Image Processing*, 17(8):1443–1451, 2008.

[12] J. Jackson, A. J. Yezzi, and S. Soatto. Dynamic shape and appearance modeling via moving and deforming layers. *International Journal of Computer Vision*, 2008.

[13] Y. Kim, A. Martínez, and A. Kak. Robust motion estimation under varying illumination. *Image and Vision Computing*, 23(4):365–375, 2005.

[14] V. Kolmogorov and R. Zabih. Computing visual correspondence with occlusions via graph cuts. In *International Conference on Computer Vision*, volume 2, pages 508–515. Citeseer, 2001.

[15] K. Lim, A. Das, and M. Chong. Estimation of occlusion and dense motion fields in a bidirectional Bayesian framework. *IEEE Transactions on Pattern Analysis and Machine Intelligence*, pages 712–718, 2002.

[16] S. Negahdaripour. Revised definition of optical flow: Integration of radiometric and geometric cues for dynamic scene analysis. *IEEE Transactions on Pattern Analysis and Machine Intelligence*, pages 961–979, 1998.

[17] Y. Nesterov. A method for unconstrained convex minimization problem with the rate of convergence O (1/k2). In *Doklady AN SSSR*, volume 269, pages 543–547, 1983.

[18] Y. Nesterov. Smooth minimization of non-smooth functions. *Mathematical Programming*, 103(1):127–152, 2005.

[19] M. Proesmans, L. Van Gool, and A. Oosterlinck. Determination of optical flow and its discontinuities using a non-linear diffusion. In *European Conference on Computer Vision*, 1994.

[20] M. Raptis, A. Ayvaci, and S. Soatto. Occlusion Detection and Motion Estimation via Convex Optimization. Technical report, UCLA CAM 10-36, June 2010.

[21] D. Shulman and J. Herve. Regularization of discontinuous flow fields. In *Proc. of Workshop on Visual Motion*, pages 81–86, 1989.

[22] S. Soatto. Steps Towards a Theory of Visual Information. Technical report, UCLA-CSD100028, September 2010.

[23] S. Soatto, A. J. Yezzi, and H. Jin. Tales of shape and radiance in multiview stereo. In *Intl. Conf. on Comp. Vision*, pages 974–981, October 2003.

[24] C. Strecha, R. Fransens, and L. Van Gool. A probabilistic approach to large displacement optical flow and occlusion detection. In *ECCV Workshop SMVP*, pages 71–82. Springer, 2004.

[25] J. Sun, Y. Li, S. Kang, and H. Shum. Symmetric stereo matching for occlusion handling. In *IEEE Conference on Computer Vision and Pattern Recognition*, volume 2, page 399, 2005.

[26] C. Teng, S. Lai, Y. Chen, and W. Hsu. Accurate optical flow computation under non-uniform brightness variations. *Computer vision and image understanding*, 97(3):315–346, 2005.

[27] R. Tibshirani. Regression shrinkage and selection via the lasso. *Journal of the Royal Statistical Society. Series B (Methodological)*, 58(1):267–288, 1996.

[28] A. Verri and T. Poggio. Motion field and optical flow: Qualitative properties. *IEEE Transactions on Pattern Analysis and Machine Intelligence*, 11(5):490–498, 1989.

[29] A. Wedel, T. Pock, C. Zach, H. Bischof, and D. Cremers. An improved algorithm for TV-L1 optical flow. In *Statistical and Geometrical Approaches to Visual Motion Analysis: International Dagstuhl Seminar*. Springer, 2009.

[30] J. Xiao, H. Cheng, H. Sawhney, C. Rao, M. Isnardi, et al. Bilateral filtering-based optical flow estimation with occlusion detection. *Lecture Notes in Computer Science*, 3951:211, 2006.

